# Tackling Uncertain Correspondences for Multi-Modal Entity Alignment

**Liyi Chen[1], Ying Sun[2], Shengzhe Zhang[1], Yuyang Ye[3], Wei Wu[1], Hui Xiong[2,4]***
[1] University of Science and Technology of China, [2] Thrust of Artificial Intelligence,
The Hong Kong University of Science and Technology (Guangzhou),
[3] Rutgers University, [4] Department of Computer Science and Engineering,
The Hong Kong University of Science and Technology
liyichencly@gmail.com, {owen_zsz, urara}@mail.ustc.edu.cn,
yings@hkust-gz.edu.cn, yuyang.ye@rutgers.edu, xionghui@ust.hk

## Abstract

Recently, multi-modal entity alignment has emerged as a pivotal endeavor for the integration of Multi-Modal Knowledge Graphs (MMKGs) originating from diverse data sources. Existing works primarily focus on fully depicting entity features by designing various modality encoders or fusion approaches. However, uncertain correspondences between inter-modal or intra-modal cues, such as weak inter-modal associations, description diversity, and modality absence, still severely hinder the effective exploration of aligned entity similarities. To this end, in this paper, we propose a novel Tackling uncertain correspondences method for Multi-modal Entity Alignment (TMEA). Specifically, to handle diverse attribute knowledge descriptions, we design alignment-augmented abstract representation that incorporates the large language model and in-context learning into attribute alignment and filtering for generating and embedding the attribute abstract. In order to mitigate the influence of the modality absence, we propose to unify all modality features into a shared latent subspace and generate pseudo features via variational autoencoders according to existing modal features. Then, we develop an inter-modal commonality enhancement mechanism based on cross-attention with orthogonal constraints, to address weak semantic associations between modalities. Extensive experiments on two real-world datasets validate the effectiveness of TMEA with a clear improvement over competitive baselines.

## 1 Introduction

Multi-Modal Knowledge Graphs (MMKGs) effectively store a substantial volume of knowledge encompassing diverse modalities including visual, relational, and attribute information in a structured and organized manner [23, 43, 44]. They greatly facilitate the advancement of various downstream tasks such as recommender systems [41, 47, 12, 34], video understanding [40, 51, 39], and domain-specific applications [7, 30, 35, 49]. In order to construct a more comprehensive multi-modal knowledge base for supporting external knowledge comprehension and reasoning, the demand for integrating MMKGs from heterogeneous data sources has become extremely urgent. As a result, multi-modal entity alignment to match entities referring to identical real-world concepts from distinct MMKGs has emerged as an increasingly critical task for their integration [23, 5, 6].

In the literature, early investigations into entity alignment primarily rely on translational models [9, 53, 31] or graph neural networks [36, 3] to capture semantic relatedness of aligned entities through the geometrical structures in an embedding space. However, these methods were originally devised for

---

traditional knowledge graphs and struggle to accommodate diverse multi-modal knowledge, which makes it difficult to fulfill the need for aligning entities across MMKGs. Therefore, recent studies [5, 21, 19] have introduced diverse modality encoders or fusion approaches tailored to the characteristics of multi-modal knowledge, striving to fully extract information from all modalities so as to effectively portray the features of entities. For modality encoders, MSNEA [6] develops vision-guided encoders for relational and attribute knowledge to achieve inter-modal enhancement. ACK-MMEA [19] unifies the encoding of multi-modal fea-

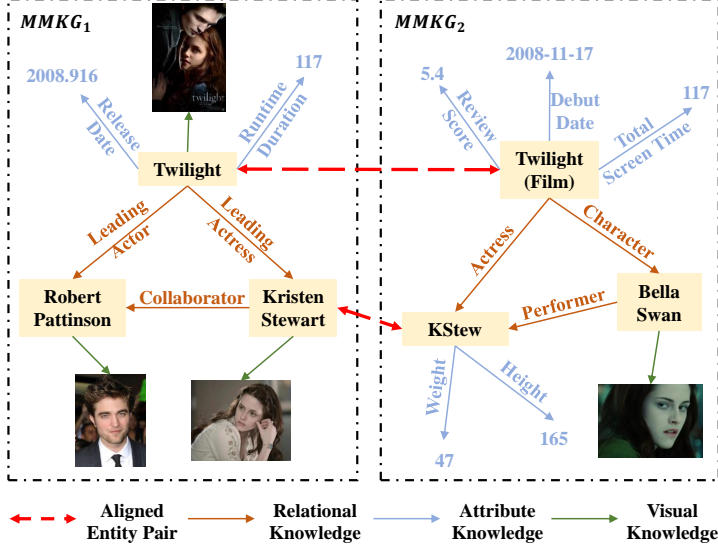

Figure 1: A toy example to illustrate the uncertain correspondences in multi-modal entity alignment, where entities are highlighted with a yellow background.

tures by designing a specialized graph neural network to ensure consistent modal aggregation. Regarding fusion approaches, MMEA [5] synthesizes multiple representations by transferring each uni-modal embedding into a common space. EVA [22] incorporates multi-modal features into a combined embedding using an attention-driven modality weighting mechanism.

Although these methods have made remarkable progress in extracting and fusing multi-modal features for entity representation, there still remain many substantial challenges, particularly the limitation of uncertain correspondences between inter-modal or intra-modal cues of entities [33, 16]. Figure 1 illustrates a toy example of such uncertain correspondences. Firstly, the weak semantic associations among diverse modal knowledge of entities pose a considerable challenge in inter-modal enhancement. In $MMKG_1$, "Twilight" has attributes "Release Date" and "Runtime Duration" along with an image of a movie poster containing two roles. The poster does not include content that could semantically overlap with these attributes, which makes it hard to capture their feature commonality to perform inter-modal enhancement. Secondly, the distinct descriptive manners of entities across different MMKGs complicate the matching of attributes with the same meanings between aligned entities. For instance, "Release Date" for "Twilight" in $MMKG_1$ and "Debut Date" for "Twilight (film)" in $MMKG_2$ both mean the initial release date of the movie. This requires a more profound semantic understanding to ascertain if they convey the same meanings. Thirdly, MMKGs often lack complete entity knowledge, with certain modalities entirely devoid of data. This impedes the differentiation of whether two entities are aligned, as this distinction can be readily misled by consistent modality features of unaligned entities. Due to missing attributes for "Kristen Stewart" in $MMKG_1$ and images for "KStew" in $MMKG_2$, there is a high chance of erroneous matching "Bella Swan" in $MMKG_2$ based on its visual similarity to "Kristen Stewart". Given the challenges presented above, the exploration of tackling uncertain correspondences is crucial in multi-modal entity alignment task.

To address these challenges, in this paper, we propose a novel **T**ackling uncertain correspondences method for **M**ulti-modal **E**ntity **A**lignment (**TMEA**). Specifically, we encode relational, attribute, and visual knowledge into their preliminary feature representations in a Multi-modal Knowledge Encoder (MKE) module. In MKE, to handle diverse attribute knowledge descriptions, we particularly design alignment-augmented abstract representation that incorporates the Large Language Model [25] (LLM) and in-context learning [13] into attribute alignment and filtering for generating and embedding the attribute abstract. To mitigate the impact of the modality absence, we devise a Missing Modality Imputation (MMI) module to complete the missing modality by unifying diverse modalities into a shared latent subspace and generating a pseudo feature via Variational AutoEncoders (VAEs) [18] according to existing modal features. For the purpose of addressing the weak semantic associations between modalities, we develop an inter-modal commonality enhancement mechanism based on cross-attention with orthogonal constraints in another Multi-modal Commonality Enhancement (MCE)

module. Extensive experiments on two real-world datasets verify the effectiveness and superiority of TMEA with an obvious improvement over competitive baselines [1]. The main contributions of this paper are summarized as follows:

- In this paper, we focus on tackling uncertain correspondences between inter-modal or intra-modal cues of entities for multi-modal entity alignment, including weak inter-modal associations, description diversity, and modality missing.

- We propose a novel TMEA model to address these three issues, by specially designing the inter-modal commonality enhancement mechanism, alignment-augmented abstract representation, and pseudo feature generation.

- We conduct extensive experiments on two real-world datasets, FB15K-DB15K and FB15K-YG15K. The experimental results clearly demonstrate the effectiveness and superiority of TMEA, with at least a 32.8% increase in Hits@1 over competitive baselines.

## 2 Related Work

In this section, we introduce the related research categorized into traditional entity alignment and multi-modal entity alignment.

**Traditional Entity Alignment.** Traditional knowledge graphs are composed of textual symbol triples, which often result in information overload [4]. Therefore, entity alignment was proposed to match the identical entities in various KGs for their integration [9]. Existing methods mainly rely on entity embedding techniques, which can be grouped into triple-based embedding [53, 31] and neighbor-based embedding [36, 42]. *Triple-based embedding* defines an energy function to measure the plausibility of triples. For example, MTransE [9] adopted TransE [2] as the entity encoder to embed different KGs into independent vector spaces and then learned the transitions between aligned entities. IPTransE [53] implemented an iterative strategy and employed it to supplement probably aligned entities into the training set. *Neighbor-based embedding* exploits the subgraph structure organized by partial relations between entities. For instance, GCN-Align [36] used graph convolutional networks to aggregate the neighborhood information of entities. MixTEA [42] instructed the graph model learning by an end-to-end mixture teaching of manually labeled mappings and probabilistic pseudo mappings. PEEA [32] increased the connections between far-away entities and labeled ones by incorporating positional information into the representation learning with a position attention layer. However, they cannot accommodate diverse multi-modal knowledge and satisfy the requirement for multi-modal entity alignment.

**Multi-Modal Entity Alignment.** Due to the urgent demand for the integration of MMKGs, extensive research has been dedicated to multi-modal entity alignment [23, 6, 21, 19]. Initially, multi-modal entity alignment was regarded as a special type of link prediction task, aiming to connect entities through the "SameAs" relation [23]. Subsequently, the first attempt in multi-modal entity alignment introduced the pioneering MMEA [5] approach, which transferred multi-modal representations into a unified space by minimizing the distance between the fused embeddings and each uni-modal embedding. Later, Liu et al. [22] believed that visual modality has abundant features as the advantage, thus leveraging visual similarities to augment the training data by iteratively adding aligned pairs. The previous studies employed independent encoders for feature extraction of each modality, without considering interactions between different modalities. Therefore, MSNEA [6] proposed a modality interaction mechanism to guide relationship learning and attribute selection using visual features, and designed multi-modal contrastive learning to make features of aligned entities more similar. Owing to the impact of contextual gaps on effectiveness, ACK-MMEA [19] utilized merge and generate operators to build an attribute-consistent MMKG, and aggregated the consistent information through graph neural networks. Despite their remarkable advancement, there still remain some significant challenges like uncertain correspondences between inter-modal or intra-modal cues of entities.

## 3 Preliminary

**Multi-Modal Knowledge Graph (MMKG).** A MMKG consists of relational, attribute, and visual knowledge and is formally defined as $\mathcal{G} = (\mathcal{E}, \mathcal{R}, \mathcal{I}, \mathcal{A}, \mathcal{V}, \mathcal{T}, \mathcal{P})$. Here, $\mathcal{E}$, $\mathcal{R}$, $\mathcal{I}$, $\mathcal{A}$, $\mathcal{V}$, $\mathcal{T}$, $\mathcal{P}$ denote the sets of entities, relations, images, attributes, attribute values, triples, and entity-image

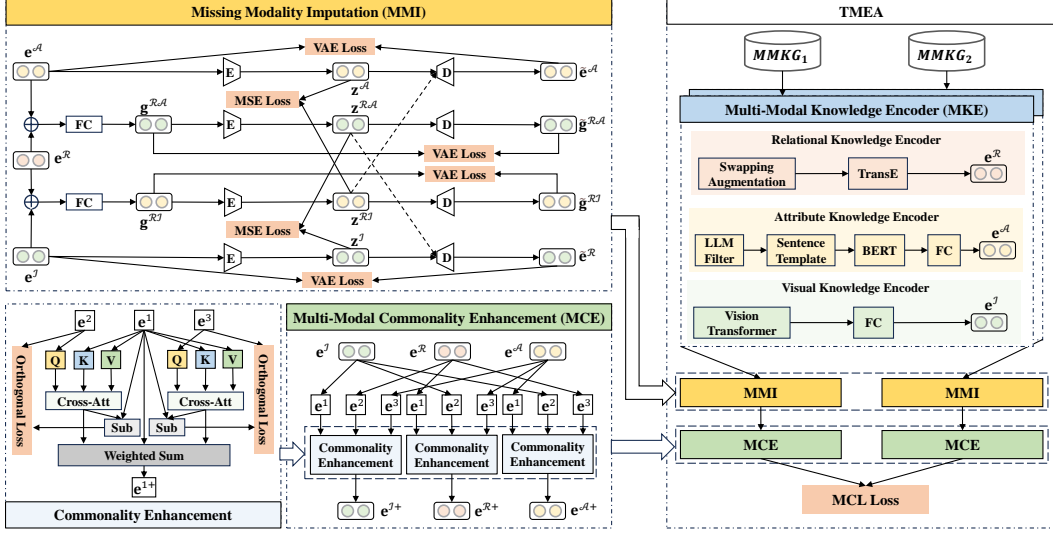

Figure 2: The framework overview of TMEA.

pairs, respectively. Specifically, the triple set $\mathcal{T}$ includes relational and attribute triples, denoted as $\mathcal{T} = \mathcal{T}_\mathcal{R} \cup \mathcal{T}_\mathcal{A}$, where $\mathcal{T}_\mathcal{R} = \{(e_h, r, e_t) \mid e_h, e_t \in \mathcal{E}, r \in \mathcal{R}\}$ refers to the set of relational triples, and $\mathcal{T}_\mathcal{A} = \{(e, a, v) \mid e \in \mathcal{E}, a \in \mathcal{A}, v \in \mathcal{V}\}$ represents the set of attribute triples. $\mathcal{P} = \{(e, i) \mid e \in \mathcal{E}, i \in \mathcal{I}\}$ indicates that the image $i$ describes the entity $e$.

**Multi-Modal Entity Alignment (MMEA).** We consider the MMEA task between two MMKGs $\mathcal{G}_1$ and $\mathcal{G}_2$. The objective of MMEA is to pair 1-to-1 corresponding entities that describe identical concepts in the real world but exist in separate MMKGs. Represented symbolically, the inputs are two MMKGs $\mathcal{G}_1 = (\mathcal{E}_1, \mathcal{R}_1, \mathcal{I}_1, \mathcal{A}_1, \mathcal{V}_1, \mathcal{T}_1, \mathcal{P}_1)$ and $\mathcal{G}_2 = (\mathcal{E}_2, \mathcal{R}_2, \mathcal{I}_2, \mathcal{A}_2, \mathcal{V}_2, \mathcal{T}_2, \mathcal{P}_2)$, and the output is the set of aligned entity pairs $\mathcal{S} = \{(e_1, e_2) \mid e_1 \in \mathcal{E}_1, e_2 \in \mathcal{E}_2, e_1 \equiv e_2\}$, where $\equiv$ means the equivalence of two entities.

## 4 Methodology

In this section, we introduce the technical details of our proposed TMEA. As illustrated in Figure 2, TMEA comprises three main modules: 1) *Multi-modal Knowledge Encoder* (MKE) module to encode relational, attribute, and visual knowledge into their preliminary feature representations, with alignment-augmented abstract representation to address diverse attribute knowledge descriptions; 2) *Missing Modality Imputation* (MMI) module to unify diverse modalities into a shared latent subspace and generate a pseudo feature via VAEs according to existing modal features for completing the missing modality; 3) *Multi-modal Commonality Enhancement* (MCE) module to enhance semantic associations between modalities by designing an inter-modal commonality enhancement mechanism based on cross-attention with orthogonal constraints. For model optimization strategy, we integrate multi-modal contrastive learning and bi-directional iteration.

### 4.1 Multi-Modal Knowledge Encoder

In this module, we encode the relational, attribute, and visual modalities of knowledge to acquire their preliminary feature representations. In particular, to address the diversity of descriptive manners of attribute knowledge, we design alignment-augmented abstract representation that incorporates the large language model and in-context learning into attribute alignment and filtering for generating and embedding the attribute abstract.

**Relational Knowledge Encoder.** Relational knowledge is composed of a set of relational triples, in the form of $(e_h, r, e_t) \in \mathcal{T}_\mathcal{R}$. Here, we take TransE [2] with the swapping strategy [5] as the relational knowledge encoder to interpret $r$ as a translation vector from $e_h$ to $e_t$, making relationships captured by simple geometric operations. The score function $f(e_h, r, e_t)$ and the margin-based loss $\mathcal{L}_{TranE}$ are defined as follows:

$$f(e_h, r, e_t) = \left\| \mathbf{e}_h^\mathcal{R} + \mathbf{r} - \mathbf{e}_t^\mathcal{R} \right\|_2^2, \tag{1}$$

$$\mathcal{L}_{TranE} = \sum_{\tau \in \mathcal{T}_{\mathcal{R}}} \sum_{\tau^- \in \mathcal{T}_{\mathcal{R}}^-} \max \left( 0, \gamma + f\left( \tau \right) - f\left( \tau^- \right) \right), \tag{2}$$

where $\mathbf{e}_h^{\mathcal{R}}$, $\mathbf{e}_t^{\mathcal{R}}$ are initial relational feature vectors of $e_h$ and $e_t$, $\mathbf{r}$ is the feature embedding of $r$, $\|\cdot\|_2$ is the $L_2$ norm, $\gamma$ is the margin hyperparameter , and $\mathcal{T}_{\mathcal{R}}^-$ is the set of negative examples. By this means, we can obtain the initial relational feature of the entity, denoted as $\mathbf{e}^{\mathcal{R}}$.

**Attribute Knowledge Encoder.** An attribute triple can be denoted as $(e, a, v) \in \mathcal{T}_{\mathcal{A}}$. Aligning attributes across different MMKGs is challenging due to the variations in the expression of identical attributes. It necessitates more than a basic lexical comparison and requires a profound semantic understanding to ascertain whether attributes with different labels indeed represent the same concept. Recently, the powerful semantic understanding capabilities of LLMs have been proven across multiple tasks [1, 8, 10]. The pre-existing knowledge within LLMs can help bridge gaps in information that may not be explicitly stated. Moreover, in-context learning [37, 52] is a prompt engineering method where task demonstrations are included in the prompt. With in-context learning, off-the-shelf LLM can solve attribute alignment without fine-tuning. Therefore, we design an alignment-augmented abstract representation that incorporates the GPT 3.5 [25] and in-context learning [13] into attribute alignment and filtering for generating and embedding the attribute abstract. For attribute alignment, the prompt is devised as shown in Appendix A. As the attributes that cannot be aligned often become noise [45], we only retain those attributes that can be aligned according to the results of GPT 3.5. Next, we convert the filtered attribute triples into a textual abstract sentence. For an entity $e$ and its $n$ filtered attribute triples $(e, a_1, v_1), (e, a_2, v_2), ..., (e, a_n, v_n)$, the template of abstract sentence $S$ is "$a_1$ is $v1$, $a_2$ is $v2$, ..., $a_n$ is $v_n$". Given two attribute triples ("Twilight", "Release Date", "2008.916") and ("Twilight", "Runtime Duration", "117") related to "Twilight", we construct a abstract sentence of "Release Date is 2008.916, Runtime Duration is 117". Then, we use a pre-trained BERT [11] model to generate the representation of this sentence, thus forming the attribute embedding of the entity in the following:

$$\mathbf{e}^{\mathcal{A}} = \mathbf{W}_{\mathcal{A}} \frac{1}{n} \sum_{j=1}^{n} BERT(w_j) + \mathbf{b}_{\mathcal{A}}, \tag{3}$$

where $w_j$ denotes the $j$-th word in $S$, $BERT(\cdot)$ represents the hidden feature vector in the last layer of BERT, $\mathbf{W}_{\mathcal{A}}$ is the weight matrix, and $\mathbf{b}_{\mathcal{A}}$ is the bias vector.

**Visual Knowledge Encoder.** Visual knowledge is typically comprised of entity-image pairs, denoted as $(e, i) \in \mathcal{P}$. We employ a pre-trained Vision Transformer (ViT) [14] to extract image features. Specifically, the last fully connected layer and softmax layer are removed to acquire the image embeddings of entities. The initial visual feature embedding $\mathbf{e}^{\mathcal{I}}$ for $(e, i)$ is generated as follows:

$$\mathbf{e}^{\mathcal{I}} = \mathbf{W}_{\mathcal{I}} ViT(i) + \mathbf{b}_{\mathcal{I}}, \tag{4}$$

where $ViT(\cdot)$ is the backbone of ViT, and $\mathbf{W}_{\mathcal{I}}$, $\mathbf{b}_{\mathcal{I}}$ are the weight matrix and bias vector.

### 4.2 Missing Modality Imputation

In real-world scenarios, entities in MMKGs often lack the knowledge of certain modalities. This omission can result from limitations in data collection methods or source availability. Incomplete modal knowledge will potentially lead to erroneous alignment. Effectively addressing and compensating for these missing modalities to guarantee accurate integration are able to improve the robustness of entity alignment process.

In this module, we intend to complete the missing modality by unifying the diverse modalities into a shared latent subspace and generating a pseudo feature via Variational AutoEncoders (VAEs) [18, 28] according to existing modal features. VAEs can learn latent representations of data, capturing essential structure and distribution. We utilize VAEs to unify features of one target modality and concatenated features of other modalities into a shared latent subspace. In this latent subspace, data from various modalities become comparable. We can use the latent representation of concatenated features derived from other modalities to approximate that of the target modality, thereby generating pseudo-features.

Typically, entities in MMKGs cannot lack relational knowledge, because entities without relations are isolated nodes, rendering them devoid of the graph's interconnected significance. Therefore, we aim to complete the visual and attribute knowledge for entities. Specifically, we first reduce the dimension of concatenated features of other modalities as follows:

$$\mathbf{g}^{\mathcal{RI}} = \mathbf{W}_{ga} Concat(\mathbf{e}^{\mathcal{R}}, \mathbf{e}^{\mathcal{I}}) + \mathbf{b}_{ga}, \tag{5}$$

$$\mathbf{g}^{\mathcal{R}\mathcal{A}} = \mathbf{W}_{gi} Concat(\mathbf{e}^{\mathcal{R}}, \mathbf{e}^{\mathcal{A}}) + \mathbf{b}_{gi}, \tag{6}$$

where $Concat(\cdot)$ is the concatenation operation, $\mathbf{W}_{g*}$, $\mathbf{b}_{g*}$ are projection matrix and bias vector. Then, we use multiple VAEs to learn the latent representations. For each VAE, $\mathbf{x} \in \mathbf{X}$ is a feature of target modality, $\mathbf{z} \in \mathbf{Z}$ is the representation of $\mathbf{x}$ in the latent space, $q_\phi(\mathbf{z}|\mathbf{x})$ is a probabilistic encoder, and $p_\theta(\mathbf{x}|\mathbf{z})$ is a probabilistic decoder. The loss of VAEs is designed as follows:

$$\mathcal{L}_{VAE}(\mathbf{X}) = \mathbb{E}_{(\mathbf{x},\mathbf{z}) \sim q_\phi(\mathbf{Z}|\mathbf{X})} \left[ \log p_\theta(\mathbf{x}|\mathbf{z}) \right] - D_{KL}\left( q_\phi(\mathbf{z}|\mathbf{x}) \| p_\theta(\mathbf{z}) \right), \tag{7}$$

$$\mathcal{L}_v = \mathcal{L}_{VAE}(\mathbf{E}^{\mathcal{A}}) + \mathcal{L}_{VAE}(\mathbf{G}^{\mathcal{R}\mathcal{I}}) + \mathcal{L}_{VAE}(\mathbf{E}^{\mathcal{I}}) + \mathcal{L}_{VAE}(\mathbf{G}^{\mathcal{R}\mathcal{A}}), \tag{8}$$

where $D_{KL}(\cdot)$ is the KL divergence, $\mathbf{E}^*$ is the set of entity features of modality $*$, and $\mathbf{G}^{\mathcal{R}*}$ is the set of concatenated features of relation and modality $*$. To unify features of one target modality and concatenated features of other modalities into a shared latent subspace, we minimize the distance of their latent representations as follows:

$$\mathcal{L}_{mse} = MSE(\mathbf{Z}^{\mathcal{A}}, \mathbf{Z}^{\mathcal{R}\mathcal{I}}) + MSE(\mathbf{Z}^{\mathcal{I}}, \mathbf{Z}^{\mathcal{R}\mathcal{A}}), \tag{9}$$

where $MSE(\cdot)$ is the mean-square error. Through this operation, we can use $\mathbf{Z}^{\mathcal{R}\mathcal{I}}$ to approximate $\mathbf{Z}^{\mathcal{A}}$, and $\mathbf{Z}^{\mathcal{R}\mathcal{A}}$ to approximate $\mathbf{Z}^{\mathcal{I}}$. In the generation stage, we exploit the results of VAEs to complete the missing modality. We input the latent representation $\mathbf{z}^{\mathcal{R}\mathcal{I}}$, $\mathbf{z}^{\mathcal{R}\mathcal{A}}$ into the decoders $p_\theta(\mathbf{e}^{\mathcal{A}}|\mathbf{z}^{\mathcal{A}})$, $p_\theta(\mathbf{e}^{\mathcal{I}}|\mathbf{z}^{\mathcal{I}})$ to generate a pseudo feature $\widetilde{\mathbf{e}}^{\mathcal{A}}$, $\widetilde{\mathbf{e}}^{\mathcal{I}}$, respectively.

### 4.3 Multi-Modal Commonality Enhancement

Generally, tenuous semantic associations between disparate modal representations of knowledge significantly hamper efforts toward inter-modal enhancement. In multi-modal tasks, semantic associations between different modalities are often utilized to mutually enhance features across modalities [50, 27]. If the semantic association is weak, mechanisms designed for interactions between different modalities may negatively impact overall performance, potentially resulting in performance worse than that of a single modality.

In this module, we enhance semantic associations between modalities by specially designing an inter-modal commonality enhancement mechanism based on cross-attention with orthogonal constraints, thereby facilitating a more holistic understanding of the entity. Cross-attention architectures have been empirically substantiated to efficiently capture semantic associations between multiple modalities [38, 29]. Specifically, we consider two distinct modal features, denoted as $\mathbf{x}$ and $\mathbf{y}$. $\mathbf{x}$ is designated as the query, while $\mathbf{y}$ assumes the roles of both key and value within the attention. The attention mechanism computes a weighted sum over $\mathbf{y}$, prioritizing elements that are most relevant to $\mathbf{x}$. This process refines $\mathbf{y}$ based on the commonality with $\mathbf{x}$. The attention is defined as follows:

$$Att(\mathbf{Q}, \mathbf{K}, \mathbf{V}) = softmax\left( \frac{\mathbf{Q}\mathbf{K}^T}{\sqrt{d}} \right) \mathbf{V}, head_j = Att\left( \mathbf{x}\mathbf{W}_j^Q, \mathbf{y}\mathbf{W}_j^K, \mathbf{y}\mathbf{W}_j^V \right), \tag{10}$$

$$MH(\mathbf{x}, \mathbf{y}) = Concat\left( head_1, \ldots, head_\eta \right) \mathbf{W}_h + \mathbf{b}_h, \tag{11}$$

where $\mathbf{Q}, \mathbf{K}, \mathbf{V}$ denote the query, key and value respectively, $softmax(\cdot)$ is the activation function, $d$ is the dimension of key, $\eta$ is the number of heads, $\mathbf{W}_j^Q, \mathbf{W}_j^K, \mathbf{W}_j^V$ are projection matrices for $j$-th attention head, and $\mathbf{W}_h, \mathbf{b}_h$ are the projection matrix and bias vector.

Building upon this cross-attention, we concurrently extract the commonalities among relational, attribute, and visual modalities to enhance the overall representation. The operation is as follows:

$$\mathbf{e}^{\mathcal{R}+} = w_{\mathcal{R}\mathcal{R}}\mathbf{e}^{\mathcal{R}} + w_{\mathcal{R}\mathcal{I}}MH_{\mathcal{R}\mathcal{I}}(\mathbf{e}^{\mathcal{I}}, \mathbf{e}^{\mathcal{R}}) + w_{\mathcal{R}\mathcal{A}}MH_{\mathcal{R}\mathcal{A}}(\mathbf{e}^{\mathcal{A}}, \mathbf{e}^{\mathcal{R}}), \tag{12}$$

$$\mathbf{e}^{\mathcal{A}+} = w_{\mathcal{A}\mathcal{A}}\mathbf{e}^{\mathcal{A}} + w_{\mathcal{A}\mathcal{I}}MH_{\mathcal{A}\mathcal{I}}(\mathbf{e}^{\mathcal{I}}, \mathbf{e}^{\mathcal{A}}) + w_{\mathcal{A}\mathcal{R}}MH_{\mathcal{A}\mathcal{R}}(\mathbf{e}^{\mathcal{R}}, \mathbf{e}^{\mathcal{A}}), \tag{13}$$

$$\mathbf{e}^{\mathcal{I}+} = w_{\mathcal{I}\mathcal{I}}\mathbf{e}^{\mathcal{I}} + w_{\mathcal{I}\mathcal{R}}MH_{\mathcal{I}\mathcal{R}}(\mathbf{e}^{\mathcal{R}}, \mathbf{e}^{\mathcal{I}}) + w_{\mathcal{I}\mathcal{A}}MH_{\mathcal{I}\mathcal{A}}(\mathbf{e}^{\mathcal{A}}, \mathbf{e}^{\mathcal{I}}), \tag{14}$$

where $w_{*\#}$, $*, \# \in \{\mathcal{R}, \mathcal{A}, \mathcal{I}\}$ are learnable weights normalized by the Softmax, and $MH_{*\#}(\mathbf{e}^{\#}, \mathbf{e}^{*})$ is the result of multi-head cross-attention for commonality extraction between two modalities.

To amplify the commonalities among features, we additionally design an orthogonal constraint loss, which ensures the difference calculated by subtracting the commonality features from the original features, is dissimilar to the query features. For example, $MH_{\mathcal{R}\mathcal{I}}(\mathbf{e}^{\mathcal{I}}, \mathbf{e}^{\mathcal{R}})$ means the refined version of $\mathbf{e}^{\mathcal{R}}$ highly relevant to $\mathbf{e}^{\mathcal{I}}$. The difference can be calculated as $\mathbf{e}^{\mathcal{R}} - MH_{\mathcal{R}\mathcal{I}}(\mathbf{e}^{\mathcal{I}}, \mathbf{e}^{\mathcal{R}})$,

representing the content within $\mathbf{e}^{\mathcal{R}}$ that does not include any elements related to $\mathbf{e}^{\mathcal{I}}$. Hence, it exhibits an orthogonal relationship with $\mathbf{e}^{\mathcal{I}}$. The orthogonal constraint loss is defined as follows:

$$\mathcal{F}_{orth}(\mathbf{x}, \mathbf{y}) = (\mathbf{x}\mathbf{y}^T)^2, \mathcal{F}_d(\mathbf{e}^*, \mathbf{e}^{\#}) = \mathbf{e}^* - MH_{*\#}(\mathbf{e}^{\#}, \mathbf{e}^*), *, \# \in \{\mathcal{R}, \mathcal{A}, \mathcal{I}\}, \quad (15)$$

$$\begin{aligned}\mathcal{L}_o =&\mathcal{F}_{orth}(\mathcal{F}_d(\mathbf{e}^{\mathcal{R}}, \mathbf{e}^{\mathcal{I}}), \mathbf{e}^{\mathcal{I}}) + \mathcal{F}_{orth}(\mathcal{F}_d(\mathbf{e}^{\mathcal{R}}, \mathbf{e}^{\mathcal{A}}), \mathbf{e}^{\mathcal{A}}) + \mathcal{F}_{orth}(\mathcal{F}_d(\mathbf{e}^{\mathcal{A}}, \mathbf{e}^{\mathcal{R}}), \mathbf{e}^{\mathcal{R}})+ \\ &\mathcal{F}_{orth}(\mathcal{F}_d(\mathbf{e}^{\mathcal{A}}, \mathbf{e}^{\mathcal{I}}), \mathbf{e}^{\mathcal{I}}) + \mathcal{F}_{orth}(\mathcal{F}_d(\mathbf{e}^{\mathcal{I}}, \mathbf{e}^{\mathcal{R}}), \mathbf{e}^{\mathcal{R}}) + \mathcal{F}_{orth}(\mathcal{F}_d(\mathbf{e}^{\mathcal{I}}, \mathbf{e}^{\mathcal{A}}), \mathbf{e}^{\mathcal{A}}),\end{aligned} \quad (16)$$

where $\mathcal{F}_{orth}(\cdot)$ is the orthogonal constraint function, $\mathcal{F}_d(\cdot)$ is the calculation function of difference, $\mathcal{L}_o$ is designed orthogonal constraint loss for commonality enhancement.

In this way, we can accentuate the commonalities between modalities to distill inter-modal correspondences for a more synergistic and semantically coherent representation.

## 4.4 Model Optimization

Through the multi-modal knowledge learning in the prior three modules, we can obtain enhanced feature embeddings for relational, visual, and attribute modalities. By directly concatenating the enhanced features, we generate an integrated multi-modal representation of an entity as follows:

$$\mathbf{e}^{\mathcal{M}} = Concat(\mathbf{e}^{\mathcal{R}+}, \mathbf{e}^{\mathcal{I}+}, \mathbf{e}^{\mathcal{A}+}), \quad (17)$$

where $\mathbf{e}_{\mathcal{M}}$ is the holistic enhanced entity embedding of entity $e$.

To ensure that features of aligned entities from different MMKGs are more closely in the space and make the overall modeling more robust, we employ the multi-modal contrastive learning [6, 48] as our main optimization objective. Multi-modal contrastive learning contrasts the feature similarity of positive entity pairs with negative entity pairs, from multiple perspectives of holistic multi-modal representations and each uni-modal feature representation. Positive entity pairs refer to two aligned entities, and negative entity pairs refer to two unaligned entities. The main objective of optimization is defined as follows:

$$\mathcal{L}_{cl}(\mathbf{E}_1^*, \mathbf{E}_2^*) = \frac{1}{2|H|} \sum_{(e_1, e_2) \in H} (1-y)d^2(\mathbf{e}_1^*, \mathbf{e}_2^*) + y \max(\gamma_{cl} - d(\mathbf{e}_1^*, \mathbf{e}_2^*), 0)^2, \quad (18)$$

$$\mathcal{L}_{mcl} = \mathcal{L}_{cl}(\mathbf{E}_1^{\mathcal{M}}, \mathbf{E}_2^{\mathcal{M}}) + \mathcal{L}_{cl}(\mathbf{E}_1^{\mathcal{R}}, \mathbf{E}_2^{\mathcal{R}}) + \mathcal{L}_{cl}(\mathbf{E}_1^{\mathcal{A}}, \mathbf{E}_2^{\mathcal{A}}) + \mathcal{L}_{cl}(\mathbf{E}_1^{\mathcal{I}}, \mathbf{E}_2^{\mathcal{I}}), \quad (19)$$

where $\mathcal{L}_{mcl}$ is the main objective of optimization, $* \in \{\mathcal{M}, \mathcal{R}, \mathcal{A}, \mathcal{I}\}$ denotes the modality, $\mathbf{E}_1^*, \mathbf{E}_2^*$ are the sets of entity feature embeddings of modality $*$ for $\mathcal{G}_1$ and $\mathcal{G}_2$, $H$ is the set of sampled entity pairs including positive and negative samples, $d(\cdot, \cdot)$ is the function of cosine similarity, $y$ is the indicator of whether an entity pair can be aligned, and $\gamma_{cl}$ is the margin hyperparameter.

Combining all the designed losses above, the overall objective of optimization $\mathcal{L}$ is as follows:

$$\mathcal{L} = \mathcal{L}_{TransE} + \lambda_1 \mathcal{L}_o + \mathcal{L}_{mcl} + \mathcal{L}_v + \lambda_2 \mathcal{L}_{mse}. \quad (20)$$

In training stage, we minimize $\mathcal{L}$ and update the model parameters via the backpropagation. Owing to the high cost of manual annotations and the expansive scale of MMKGs, the availability of aligned entity pairs is frequently insufficient. Therefore, we adopt a bi-directional iterative strategy [24] based on the asymmetric nature of alignment directions to boost the training. The specific algorithm of this strategy is described in Appendix B.

## 5 Experiment

In this section, we provide extensive experiments on two real-world datasets. To begin, we offer a brief description of experimental setups. Next, the experimental results and analysis are presented to verify the effectiveness of TMEA. More details and results of experiments are provided in Appendix C.

## 5.1 Experimental Setups

**Datasets.** Following previous studies [5, 20], we selected two commonly used public datasets, namely FB15K-DB15K and FB15K-YG15K [23], and allocated 20% of the aligned entity pairs as training data. These datasets contain two pairs of aligned MMKGs with relational, attribute, and visual knowledge.

Table 1: The performance of multi-modal entity alignment on two datasets. The best results are indicated in bold, and second-best results are marked with underlines.

| Method | FB15K-DB15K | | | | | FB15K-YG15K | | | | |
|---|---|---|---|---|---|---|---|---|---|---|
| | H@1 | H@5 | H@10 | MR | MRR | H@1 | H@5 | H@10 | MR | MRR |
| IPTransE | 0.040 | 0.112 | 0.173 | 387.512 | 0.086 | 0.031 | 0.095 | 0.144 | 522.235 | 0.070 |
| GCN-Align | 0.043 | 0.110 | 0.155 | 810.648 | 0.082 | 0.023 | 0.072 | 0.107 | 1109.845 | 0.053 |
| BootEA | 0.323 | 0.499 | 0.579 | 205.532 | 0.410 | 0.234 | 0.374 | 0.445 | 272.120 | 0.307 |
| SEA | 0.170 | 0.335 | 0.425 | 191.903 | 0.255 | 0.141 | 0.287 | 0.371 | 207.236 | 0.218 |
| RAC | 0.203 | 0.360 | 0.432 | 453.313 | 0.281 | 0.151 | 0.281 | 0.345 | 501.795 | 0.216 |
| PEEA | 0.143 | 0.254 | 0.299 | 1265.616 | 0.198 | 0.126 | 0.223 | 0.268 | 989.414 | 0.175 |
| MixTEA | 0.115 | 0.233 | 0.299 | 340.733 | 0.178 | 0.085 | 0.177 | 0.234 | 350.439 | 0.136 |
| PoE | 0.120 | - | 0.256 | - | 0.167 | 0.109 | - | 0.241 | - | 0.154 |
| MMEA | 0.265 | 0.451 | 0.541 | 124.807 | 0.357 | 0.234 | 0.398 | 0.480 | 147.441 | 0.317 |
| EVA | 0.556 | 0.666 | 0.716 | 139.995 | 0.609 | 0.103 | 0.217 | 0.278 | 616.789 | 0.164 |
| MSNEA | 0.653 | 0.768 | 0.812 | 54.025 | 0.708 | 0.443 | 0.626 | 0.698 | 85.074 | 0.529 |
| MCLEA | 0.441 | 0.640 | 0.710 | 84.628 | 0.534 | 0.406 | 0.579 | 0.645 | 123.394 | 0.488 |
| ACK-MMEA | 0.304 | - | 0.549 | - | 0.387 | 0.289 | - | 0.496 | - | 0.360 |
| MoAlign | 0.318 | - | 0.564 | - | 0.409 | 0.296 | - | 0.525 | - | 0.378 |
| TMEA | **0.867** | **0.929** | **0.944** | **26.343** | **0.895** | **0.818** | **0.891** | **0.916** | **32.864** | **0.853** |

**Evaluation Metrics.** We assessed the alignment probability between entities from different MMKGs based on cosine similarity calculation. Then, we chose H@N (Hits@N), MRR (Mean Reciprocal Rank), and MR (Mean Rank) as metrics to evaluate all the models. A higher value for H@N and MRR indicates better performance, whereas a lower value for MR suggests the same.

**Compared Methods** We benchmarked TMEA against a selection of both representative and state-of-the-art approaches. *For traditional KGs*: IPTransE [53], GCN-Align [36], BootEA [31], SEA [26], RAC [46], PEEA [32], and MixTEA [42]. *For MMKGs*: PoE [23], MMEA [5], EVA [22], MSNEA [6], MCLEA [21], ACK-MMEA [19], and MoAlign [20].

## 5.2 Performance Comparison

We compared TMEA with several representative and state-of-the-art baselines for multi-modal entity alignment task on FB15K-DB15K and FB15K-YG15K datasets. Table 1 shows the performance of all methods with 20% aligned entity pairs for training. Overall, our proposed TMEA achieves the best performance on both datasets. Specifically, we have the following findings. First, TMEA exhibits significant improvements with a notable margin compared to not only traditional but also multi-modal methods. Compared to the strongest traditional alignment method, BootEA, TMEA achieves at least a 168.4%, 63.0%, and 118.3% improvement in H@1, H@10, and MRR on two datasets, respectively. When assessed against the leading multi-modal method, MSNEA, TMEA attained at least a 32.8%, 16.3%, and 26.4% increase in H@1, H@10, and MRR, respectively. Besides, we test the statistical significance and the results suggest TMEA has significant improvements ($p\text{-}value < 0.001$) over baselines. Second, comparing multi-modal entity alignment methods to traditional ones, we observe a clear advantage in favor of multi-modal approaches in most cases. TMEA, MSNEA, and MCLEA surpass all traditional approaches, which suggests the effect of leveraging the rich and varied information that multi-modal knowledge provides. This additional context and modeling allow for a more comprehensive depiction of entity features, facilitating a more accurate determination of potential alignments. Third, our TMEA method, leveraging only 20% of aligned entity pairs for training, achieves a remarkable H@10 score of 0.944 and 0.916 on two datasets. This large margin of superiority over other multi-modal entity alignment methods can be attributed to the considering and effective tackling of uncertain correspondences in the alignment process. By addressing these inherent issues, TMEA is demonstrated to enhance the precision and reliability of entity alignment for MMKGs.

## 5.3 Ablation Study

To verify the influence of each part in TMEA, we conducted two sets of ablation studies, one set for modality ablation and another for component ablation. The variants for modality ablation are named w/o V, w/o R, and w/o A, corresponding to the removal of visual, relational, and attribute knowledge respectively. In terms of component ablation, w/o MMI and w/o MCE refer to the elimination of

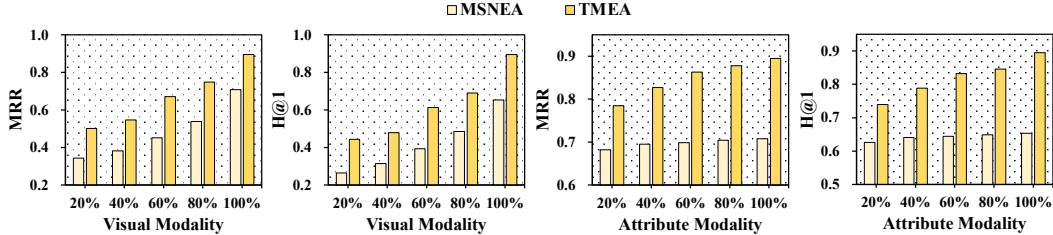

Figure 3: The performance of TMEA and the strongest baseline, MSNEA, when varying ratios of entities have visual or attribute modalities on FB15K-DB15K.

modules MMI and MCE, w/o $\mathcal{L}_o$ and w/o $\mathcal{L}_{mse}$ are the removal of constraint losses, w/o AP denotes the absence of preprocessing for attribute knowledge, and w/o IT indicates the exclusion of the iterative strategy. Table 2 shows the results of ablation study, demonstrating the effectiveness of each part in TMEA. In the results of the modality ablation study, it is evident across both datasets that each modality's features apparently contribute to the improvement of entity alignment performance. Furthermore, the most effective modality varies across datasets, which may depend on factors such as data sparsity or noise. Regarding the re-

Table 2: The ablation study to verify the impact of modality and component on two datasets.

| Method | FB15K-DB15K | | | FB15K-YG15K | | |
|---|---|---|---|---|---|---|
| | H@1 | H@10 | MR | H@1 | H@10 | MR |
| TMEA | **0.867** | **0.944** | **26.343** | **0.818** | **0.916** | **32.864** |
| w/o V | 0.446 | 0.684 | 227.984 | 0.609 | 0.772 | 139.901 |
| w/o R | 0.630 | 0.742 | 290.519 | 0.657 | 0.834 | 96.719 |
| w/o A | 0.793 | 0.904 | 52.959 | 0.569 | 0.735 | 157.695 |
| w/o AP | 0.786 | 0.903 | 45.263 | 0.593 | 0.757 | 138.562 |
| w/o MMI | 0.852 | 0.935 | 32.867 | 0.779 | 0.885 | 56.569 |
| w/o $\mathcal{L}_{mse}$ | 0.841 | 0.922 | 37.829 | 0.797 | 0.900 | 47.584 |
| w/o MCE | 0.749 | 0.868 | 48.314 | 0.636 | 0.809 | 51.573 |
| w/o $\mathcal{L}_o$ | 0.842 | 0.931 | 30.239 | 0.794 | 0.904 | 44.741 |
| w/o IT | 0.728 | 0.834 | 64.875 | 0.645 | 0.825 | 112.248 |

sults of component ablation, the effectiveness of each designed component is validated. First, the considerable drop in performance without MCE module (w/o MCE) underscores the capability of MCE module to bolster the commonality among modalities, mitigating the impact of weak semantic associations. Second, the marked deterioration in performance without attribute preprocessing (w/o AP) verifies the effect of integrating the LLM and in-context learning into filtering and abstract sentence embedding. Third, removing MMI module (w/o MMI) leads to an increase in the MR metric, and the effect is different on two datasets, indicating that the effectiveness of MMI module is contingent on the proportion of missing modalities within the dataset. Last, the absence of the iterative strategy (w/o IT) highlights the strategy's role in boosting the robustness of the model, as evidenced by a marked deterioration in MR upon its removal.

## 5.4 Modality Sensitivity Analysis

In MMKGs, entities often lack certain modalities of knowledge. Therefore, we address the missing modalities for entities in MMI module. To validate the ability of TMEA to maintain robust performance with missing modalities, we compared TMEA with the strongest baseline, MSNEA, under varying ratios of missing modal knowledge. Figure 3 presents the results of TMEA and MSNEA under scenarios with 20%, 40%, 60%, 80%, and 100% entities having visual and attribute modalities on FB15K-DB15K, respectively. We observe that TMEA outperforms MSNEA at different ratios obviously. In scenarios with missing visual modality, TMEA exhibits a slower performance decline compared to MSNEA. Moreover, when only 20% of entities have visual modality, TMEA achieves an MRR exceeding 0.5, while MSNEA falls below 0.4. For missing attribute modality, both TMEA and MSNEA show a relatively minor performance decline. With only 20% of entities having attribute modality, TMEA achieves an MRR and H@1 close to 0.8. MSNEA remains relatively unchanged across different proportions due to its insufficient attribute learning of adaptive weights, even with a substantial number of attribute features that only yield marginal improvements. This also indicates that our attribute knowledge encoder more efficiently and fully leverages the semantics of attributes.

## 5.5 Label Dependency Analysis

The availability of aligned entity pairs is frequently insufficient. Importantly, the effective and sufficient exploitation of multi-modal knowledge can significantly mitigate the dependency of methods on alignment labels during training. For the purpose of assessing these methods' reliance on the

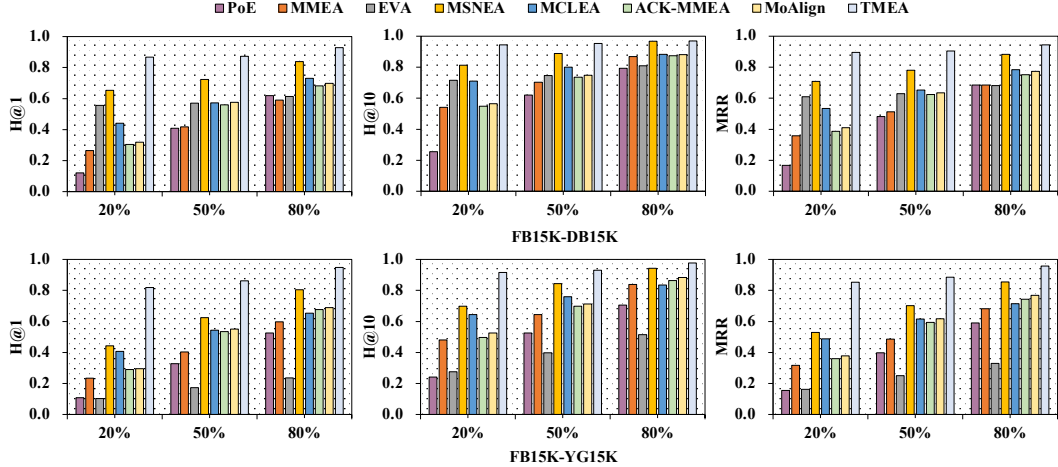

Figure 4: Comparison results with different ratios of alignment labels.

quantity of aligned entity pairs, we conducted a series of experiments to train the models using 20%, 50%, and 80% aligned entity pairs. Figure 4 shows the comparison results of all multi-modal entity alignment methods. From the overview, TMEA provides an obviously leading performance with different proportions of training data. Only TMEA achieves H@1, H@10, and MRR scores above 0.8 across various percentages. Moreover, its performance does not show a clear decline even with the reduction of training data proportions, unlike the majority of methods that experience a marked loss in performance. For instance, ACK-MMEA and MoAlign achieve lower H@10 and MRR scores than MCLEA with 50% of training data, yet surpass MCLEA when the training data is increased to 80% on FB15K-YG15K. This indicates their strong dependency on aligned entity pairs. When the aligned sample pairs reach 80%, even though the performance of TMEA and MSNEA on the H@10 metric appears to be close, there is still a noticeable difference in MRR and H@1. This suggests that MSNEA performs poorly in aligning difficult samples.

## 6   Conclusion

In this paper, we proposed a novel TMEA to tackle uncertain correspondences between inter-modal or intra-modal cues of entities for multi-modal entity alignment. Specifically, we first developed a Multi-modal Knowledge Encoder (MKE) module to encode relational, attribute, and visual knowledge into their preliminary feature representations. In MKE, to address the diversity of descriptive manners of attribute knowledge, we particularly designed alignment-augmented abstract representation that incorporated the LLM and in-context learning into attribute alignment and filtering for generating and embedding the attribute abstract. To mitigate the influence of the modality absence, we devised a Missing Modality Imputation (MMI) module to complete the missing modality by unifying diverse modalities into a shared latent subspace and generating a pseudo feature via VAEs based on existing modal features. For enhancing the semantic associations between modalities, we invented an inter-modal commonality enhancement mechanism based on cross-attention with orthogonal constraints in Multi-modal Commonality Enhancement (MCE) module. Finally, extensive experiments on two real-world datasets were conducted to verify the effectiveness of TMEA with a significant improvement over competitive baselines.

## Acknowledgments and Disclosure of Funding

This work was supported in part by the National Natural Science Foundation of China (Grant No. 923 70204), the National Key Research and Development Program of China (Grant No. 2023YFF07250 01), the Guangzhou-HKUST (GZ) Joint Funding Program (Grant No. 2023A03J0008), the Education Bureau of Guangzhou Municipality, the Guangdong Basic and Applied Basic Research Foundation (Grant No. 2024A1515011839).

## Footnotes

[1]The code is available at `https://github.com/liyichen-cly/TMEA`.

[2]https://openai.com/blog/chatgpt

[3]https://developers.google.com/freebase

[4]https://www.google.com

[5]https://www.dbpedia.org/resources/knowledge-graphs

[6]https://yago-knowledge.org/

[7] https://pytorch.org/

[8] https://github.com/OceanTangWei/PEEA

[9] https://github.com/Xiefeng69/MixTEA

[10] https://github.com/lzxlin/MCLEA

## References

[1] OpenAI: Josh Achiam, Steven Adler, Sandhini Agarwal, Lama Ahmad, Ilge Akkaya, Floren-cia Leoni Aleman, Diogo Almeida, Janko Altenschmidt, Sam Altman, Shyamal Anadkat, Red

Avila, Igor Babuschkin, Suchir Balaji, Valerie Balcom, Paul Baltescu, Haiming Bao, et al. Gpt-4 technical report, 2023.

[2] Antoine Bordes, Nicolas Usunier, Alberto Garcia-Duran, Jason Weston, and Oksana Yakhnenko. Translating embeddings for modeling multi-relational data. In *Advances in neural information processing systems*, pages 2787–2795, 2013.

[3] Yixin Cao, Zhiyuan Liu, Chengjiang Li, Juanzi Li, and Tat-Seng Chua. Multi-channel graph neural network for entity alignment. In *Proceedings of the 57th Annual Meeting of the Association for Computational Linguistics*, pages 1452–1461, 2019.

[4] Liyi Chen, Zhi Li, Weidong He, Gong Cheng, Tong Xu, Nicholas Jing Yuan, and Enhong Chen. Entity summarization via exploiting description complementarity and salience. *IEEE Transactions on Neural Networks and Learning Systems*, 34(11):8297–8309, 2023.

[5] Liyi Chen, Zhi Li, Yijun Wang, Tong Xu, Zhefeng Wang, and Enhong Chen. Mmea: Entity alignment for multi-modal knowledge graph. In *International Conference on Knowledge Science, Engineering and Management*, pages 134–147. Springer, 2020.

[6] Liyi Chen, Zhi Li, Tong Xu, Han Wu, Zhefeng Wang, Nicholas Jing Yuan, and Enhong Chen. Multi-modal siamese network for entity alignment. In *Proceedings of the 28th ACM SIGKDD conference on knowledge discovery and data mining*, pages 118–126, 2022.

[7] Liyi Chen, Chuan Qin, Ying Sun, Xin Song, Tong Xu, Hengshu Zhu, and Hui Xiong. Collaboration-aware hybrid learning for knowledge development prediction. In *Proceedings of the ACM on Web Conference 2024*, pages 3976–3985, 2024.

[8] Liyi Chen, Panrong Tong, Zhongming Jin, Ying Sun, Jieping Ye, and Hui Xiong. Plan-on-graph: Self-correcting adaptive planning of large language model on knowledge graphs. In *Proceedings of the 38th Conference on Neural Information Processing Systems*, 2024.

[9] Muhao Chen, Yingtao Tian, Mohan Yang, and Carlo Zaniolo. Multilingual knowledge graph embeddings for cross-lingual knowledge alignment. In *Proceedings of the 26th International Joint Conference on Artificial Intelligence*, page 1511–1517, 2017.

[10] Xi Chen, Xinjiang Lu, Haoran Xin, Wenjun Peng, Haoyang Duan, Feihu Jiang, Jingbo Zhou, and Hui Xiong. A table-to-text framework with heterogeneous multidominance attention and self-evaluated multi-pass deliberation. In *Findings of the Association for Computational Linguistics: EMNLP 2023*, pages 607–620, 2023.

[11] Jacob Devlin, Ming-Wei Chang, Kenton Lee, and Kristina Toutanova. Bert: Pre-training of deep bidirectional transformers for language understanding. In *Proceedings of the 2019 Conference of the North American Chapter of the Association for Computational Linguistics: Human Language Technologies*, pages 4171–4186, 2019.

[12] Leilei Ding, Dazhong Shen, Chao Wang, Tianfu Wang, Le Zhang, Hui Xiong, and Yanyong Zhang. Dgr: A general graph desmoothing framework for recommendation via global and local perspectives. In *Proceedings of the 33th International Joint Conference on Artificial Intelligence*, 2024.

[13] Qingxiu Dong, Lei Li, Damai Dai, Ce Zheng, Zhiyong Wu, Baobao Chang, Xu Sun, Jingjing Xu, and Zhifang Sui. A survey on in-context learning. *arXiv preprint arXiv:2301.00234*, 2022.

[14] Alexey Dosovitskiy, Lucas Beyer, Alexander Kolesnikov, Dirk Weissenborn, Xiaohua Zhai, Thomas Unterthiner, Mostafa Dehghani, Matthias Minderer, Georg Heigold, Sylvain Gelly, Jakob Uszkoreit, and Neil Houlsby. An image is worth 16x16 words: Transformers for image recognition at scale. In *9th International Conference on Learning Representations, ICLR 2021*, 2021.

[15] Xavier Glorot and Yoshua Bengio. Understanding the difficulty of training deep feedforward neural networks. In *Proceedings of the thirteenth international conference on artificial intelligence and statistics*, pages 249–256. JMLR Workshop and Conference Proceedings, 2010.

[16] Yatai Ji, Junjie Wang, Yuan Gong, Lin Zhang, Yanru Zhu, Hongfa Wang, Jiaxing Zhang, Tetsuya Sakai, and Yujiu Yang. Map: Multimodal uncertainty-aware vision-language pre-training model. In *Proceedings of the IEEE/CVF Conference on Computer Vision and Pattern Recognition*, pages 23262–23271, 2023.

[17] Diederik P Kingma and Jimmy Ba. Adam: A method for stochastic optimization. In *Proceedings of the 3rd International Conference on Learning Representations*, 2015.

[18] Diederik P. Kingma and Max Welling. Auto-encoding variational bayes. In *Proceedings of the 2nd International Conference on Learning Representations*, 2014.

[19] Qian Li, Shu Guo, Yangyifei Luo, Cheng Ji, Lihong Wang, Jiawei Sheng, and Jianxin Li. Attribute-consistent knowledge graph representation learning for multi-modal entity alignment. In *Proceedings of the ACM Web Conference 2023*, pages 2499–2508, 2023.

[20] Qian Li, Cheng Ji, Shu Guo, Zhaoji Liang, Lihong Wang, and Jianxin Li. Multi-modal knowledge graph transformer framework for multi-modal entity alignment. In *Findings of the Association for Computational Linguistics: EMNLP 2023*, pages 987–999, 2023.

[21] Zhenxi Lin, Ziheng Zhang, Meng Wang, Yinghui Shi, Xian Wu, and Yefeng Zheng. Multi-modal contrastive representation learning for entity alignment. In *Proceedings of the 29th International Conference on Computational Linguistics*, pages 2572–2584, 2022.

[22] Fangyu Liu, Muhao Chen, Dan Roth, and Nigel Collier. Visual pivoting for (unsupervised) entity alignment. In *Proceedings of the AAAI Conference on Artificial Intelligence (AAAI)*, 2021.

[23] Ye Liu, Hui Li, Alberto Garcia-Duran, Mathias Niepert, Daniel Onoro-Rubio, and David S Rosenblum. Mmkg: multi-modal knowledge graphs. In *European Semantic Web Conference*, pages 459–474. Springer, 2019.

[24] Xin Mao, Wenting Wang, Huimin Xu, Man Lan, and Yuanbin Wu. Mraea: An efficient and robust entity alignment approach for cross-lingual knowledge graph. In *Proceedings of the 13th International Conference on Web Search and Data Mining*, page 420–428, 2020.

[25] OpenAI. Introducing chatgpt. `https://openai.com/index/chatgpt/`, 2022. Accessed: 2022-11-30.

[26] Shichao Pei, Lu Yu, Robert Hoehndorf, and Xiangliang Zhang. Semi-supervised entity alignment via knowledge graph embedding with awareness of degree difference. In *The World Wide Web Conference*, pages 3130–3136, 2019.

[27] Leigang Qu, Meng Liu, Jianlong Wu, Zan Gao, and Liqiang Nie. Dynamic modality interaction modeling for image-text retrieval. In *Proceedings of the 44th International ACM SIGIR Conference on Research and Development in Information Retrieval*, pages 1104–1113, 2021.

[28] Dazhong Shen, Chuan Qin, Chao Wang, Hengshu Zhu, Enhong Chen, and Hui Xiong. Regularizing variational autoencoder with diversity and uncertainty awareness. In *30th International Joint Conference on Artificial Intelligence, IJCAI 2021*, pages 2964–2970. International Joint Conferences on Artificial Intelligence, 2021.

[29] Yongduo Sui, Wenyu Mao, Shuyao Wang, Xiang Wang, Jiancan Wu, Xiangnan He, and Tat-Seng Chua. Enhancing out-of-distribution generalization on graphs via causal attention learning. *ACM Transactions on Knowledge Discovery from Data*, 18(5):1–24, 2024.

[30] Ying Sun, Hengshu Zhu, Lu Wang, Le Zhang, and Hui Xiong. Large-scale online job search behaviors reveal labor market shifts amid covid-19. *Nature Cities*, 1(2):150–163, 2024.

[31] Zequn Sun, Wei Hu, Qingheng Zhang, and Yuzhong Qu. Bootstrapping entity alignment with knowledge graph embedding. In *Proceedings of the Twenty-Seventh International Joint Conference on Artificial Intelligence*, pages 4396–4402, 2018.

[32] Wei Tang, Fenglong Su, Haifeng Sun, Qi Qi, Jingyu Wang, and Hao Yang. Weakly supervised entity alignment with positional inspiration. In *Proceedings of the 16th ACM International Conference on Web Search and Data Mining*, page 814–822, 2023.

[33] Hu Wang, Jianpeng Zhang, Yuanhong Chen, Congbo Ma, Jodie Avery, Louise Hull, and G. Carneiro. Uncertainty-aware multi-modal learning via cross-modal random network prediction. In *European Conference on Computer Vision*, 2022.

[34] Shuyao Wang, Yongduo Sui, Jiancan Wu, Zhi Zheng, and Hui Xiong. Dynamic sparse learning: A novel paradigm for efficient recommendation. In *Proceedings of the 17th ACM International Conference on Web Search and Data Mining*, pages 740–749, 2024.

[35] Tianfu Wang, Liwei Deng, Chao Wang, Jianxun Lian, Yue Yan, Nicholas Jing Yuan, Qi Zhang, and Hui Xiong. Comet: Nft price prediction with wallet profiling. In *Proceedings of the 30th ACM SIGKDD Conference on Knowledge Discovery and Data Mining*, pages 5893–5904, 2024.

[36] Zhichun Wang, Qingsong Lv, Xiaohan Lan, and Yu Zhang. Cross-lingual knowledge graph alignment via graph convolutional networks. In *Proceedings of the 2018 Conference on Empirical Methods in Natural Language Processing*, pages 349–357, 2018.

[37] Jason Wei, Xuezhi Wang, Dale Schuurmans, Maarten Bosma, Fei Xia, Ed Chi, Quoc V Le, Denny Zhou, et al. Chain-of-thought prompting elicits reasoning in large language models. *Advances in Neural Information Processing Systems*, 35:24824–24837, 2022.

[38] Xi Wei, Tianzhu Zhang, Yan Li, Yongdong Zhang, and Feng Wu. Multi-modality cross attention network for image and sentence matching. In *Computer Vision and Pattern Recognition*, 2020.

[39] Shiwei Wu, Joya Chen, Kevin Qinghong Lin, Qimeng Wang, Yan Gao, Qianli Xu, Tong Xu, Yao Hu, Enhong Chen, and Mike Zheng Shou. Videollm-mod: Efficient video-language streaming with mixture-of-depths vision computation. In *Proceedings of the 38th Conference on Neural Information Processing Systems*, 2024.

[40] Shiwei Wu, Joya Chen, Tong Xu, Liyi Chen, Lingfei Wu, Yao Hu, and Enhong Chen. Linking the characters: Video-oriented social graph generation via hierarchical-cumulative gcn. In *Proceedings of the 29th ACM International Conference on Multimedia*, pages 4716–4724, 2021.

[41] Wei Wu, Chao Wang, Dazhong Shen, Chuan Qin, Liyi Chen, and Hui Xiong. Afdgcf: Adaptive feature de-correlation graph collaborative filtering for recommendations. In *Proceedings of the 47th International ACM SIGIR Conference on Research and Development in Information Retrieval*, pages 1242–1252, 2024.

[42] Feng Xie, Xin Song, Xiang Zeng, Xuechen Zhao, Lei Tian, Bin Zhou, and Yusong Tan. Mixtea: Semi-supervised entity alignment with mixture teaching. In *Findings of the Association for Computational Linguistics: EMNLP 2023*, pages 886–896, 2023.

[43] Derong Xu, Tong Xu, Shiwei Wu, Jingbo Zhou, and Enhong Chen. Relation-enhanced negative sampling for multimodal knowledge graph completion. In *Proceedings of the 30th ACM International Conference on Multimedia*, pages 3857–3866, 2022.

[44] Derong Xu, Jingbo Zhou, Tong Xu, Yuan Xia, Ji Liu, Enhong Chen, and Dejing Dou. Multi-modal biological knowledge graph completion via triple co-attention mechanism. In *2023 IEEE 39th International Conference on Data Engineering (ICDE)*, pages 3928–3941. IEEE, 2023.

[45] Hsiu-Wei Yang, Yanyan Zou, Peng Shi, Wei Lu, Jimmy Lin, and Xu Sun. Aligning cross-lingual entities with multi-aspect information. In *Proceedings of the 2019 Conference on Empirical Methods in Natural Language Processing and the 9th International Joint Conference on Natural Language Processing*, pages 4430–4440, 2019.

[46] Weixin Zeng, Xiang Zhao, Jiuyang Tang, and Changjun Fan. Reinforced active entity alignment. In *Proceedings of the 30th ACM International Conference on Information & Knowledge Management*, pages 2477–2486, 2021.

[47] He Zhang, Ying Sun, Weiyu Guo, Yafei Liu, Haonan Lu, Xiaodong Lin, and Hui Xiong. Interactive interior design recommendation via coarse-to-fine multimodal reinforcement learning. In *Proceedings of the 31st ACM International Conference on Multimedia*, pages 6472–6480, 2023.

[48] Shengzhe Zhang, Liyi Chen, Chao Wang, Shuangli Li, and Hui Xiong. Temporal graph contrastive learning for sequential recommendation. In *Proceedings of the AAAI Conference on Artificial Intelligence*, volume 38, pages 9359–9367, 2024.

[49] Weijia Zhang, Jindong Han, Zhao Xu, Hang Ni, Hao Liu, and Hui Xiong. Urban foundation models: A survey. In *Proceedings of the 30th ACM SIGKDD Conference on Knowledge Discovery and Data Mining*, pages 6633–6643, 2024.

[50] Yunhua Zhang, Hazel Doughty, and Cees Snoek. Learning unseen modality interaction. *Advances in Neural Information Processing Systems*, 36, 2024.

[51] Yuting Zhang, Zhao Zhang, Yiqing Wu, Ying Sun, Fuzhen Zhuang, Wenhui Yu, Lantao Hu, Han Li, Kun Gai, Zhulin An, et al. Tag tree-guided multi-grained alignment for multi-domain short video recommendation. In *ACM Multimedia 2024*.

[52] Denny Zhou, Nathanael Schärli, Le Hou, Jason Wei, Nathan Scales, Xuezhi Wang, Dale Schuurmans, Claire Cui, Olivier Bousquet, Quoc V Le, et al. Least-to-most prompting enables complex reasoning in large language models. In *The Eleventh International Conference on Learning Representations*, 2022.

[53] Hao Zhu, Ruobing Xie, Zhiyuan Liu, and Maosong Sun. Iterative entity alignment via joint knowledge embeddings. In *Proceedings of the Twenty-Sixth International Joint Conference on Artificial Intelligence*, pages 4258–4264, 2017.

# Appendix

## A    Prompt for Attribute Alignment

Aligning attributes necessitates a profound semantic understanding to ascertain whether attributes with different literals indeed represent the same concept. Therefore, we incorporate the GPT 3.5 [2] and in-context learning into aligning attributes from two MMKGs. The prompt used for attribute alignment is shown in Table 3.

Table 3: The prompt used for attribute alignment.

| |
| --- |
| You are an excellent attribute alignment expert, and now there are two sets of attributes in two knowledge graphs (KGs). You need to align the attributes in both KGs one by one based on whether they have the same meaning, but there may be attributes that cannot be aligned.<br>Here is an example:<br>The attributes and their values are<br>KG1: {(date of birth, 1880-10-12), (latitude, 54.20867775), (longitude, 9.70404945)},<br>KG2: {(birthDate, 1563.333), (activeYears, 1652), (birthYear, 1975), (long, 14.548)}.<br>The attributes that can be aligned are<br>(date of birth, birthDate)<br>(longitude, long)<br><br>Please align as many attributes as possible from the two KGs following. Directly output results by line, without outputting duplicate or irrelevant information. Even if there is information that cannot be aligned, do not output explanations.<br>The attributes and their values are<br>KG1: {(a1, v1), (a2, v2), ...},<br>KG2: {(a1, v1), (a2, v2), ...}. |

## B    Iterative Strategy Algorithm

We adopt a bi-directional iterative strategy [24] based on the asymmetric nature of alignment directions to boost the training. During the current iteration, if the entities $e_1 \in \mathcal{E}_1$ and $e_2 \in \mathcal{E}_2$ are the mutual nearest neighbors of each other, they will be treated as a newly aligned pair to be added into the training set for the subsequent iteration. Algorithm 1 outlines the details of this iterative process.

## C    More Experimental Details and Results

### C.1    Dataset Description

We adopted two representative public datasets commonly used in multi-modal entity alignment task [23, 6, 19], namely FB15K-DB15K and FB15K-YG15K. The statistical details of FB15K-DB15K and FB15K-YG15K are presented in Table 4. FB15K is derived from Freebase [3], a large-scale knowledge base acquired by Google [4]. It is widely applied for knowledge graph completion tasks and evaluating link prediction models. DB15K is a subset of DBpedia [5], while YG15K is a subset of YAGO [6]. The entities in FB15K can be aligned with the entities in DB15K and YG15K, respectively.

### C.2    Metric Explanation

We chose H@N (Hits@N), MRR (Mean Reciprocal Rank), and MR (Mean Rank) as evaluation metrics. According to similarity ranking, H@N counts the number of times the true aligned entity

**Algorithm 1** Bi-Directional Iterative Strategy for MMKGs

**Require:** Two MMKGs $\mathcal{G}_1$, $\mathcal{G}_2$, a set of aligned entity pairs $\mathcal{S}$, $\mathcal{E}'_1 \subseteq \mathcal{E}_1$, $\mathcal{E}'_2 \subseteq \mathcal{E}_2$ are entity sets that does not exist in $\mathcal{S}$ respectively, $\mathbf{E}_1^{\mathcal{M}}$, $\mathbf{E}_2^{\mathcal{M}}$ are sets of holistic enhanced entity embedding respectively.

**Ensure:** Model parameters $\theta$
1: Initialize model parameters $\theta$
2: **repeat**
3:     Train model on $\mathcal{G}_1$, $\mathcal{G}_2$, $\mathcal{S}$ until the loss of validation set does not decrease
4:     **for** $e \in \mathcal{E}'_1$ **do**
5:         Get the embedding $\mathbf{e}^{\mathcal{M}} \in \mathbf{E}_1^{\mathcal{M}}$ of entity $e$
6:         From $\mathcal{E}'_2$, find the entity $e'$ whose feature $\mathbf{e}'^{\mathcal{M}} \in \mathbf{E}_2^{\mathcal{M}}$ is most similar to $\mathbf{e}^{\mathcal{M}}$
7:         **if** $\mathbf{e}'^{\mathcal{M}}$ is also most similar to the embedding of entity $e$ from $\mathcal{E}'_1$ **then**
8:             $\mathcal{S} \leftarrow \mathcal{S} \cup \{(e, e')\}$
9:             $\mathcal{E}'_1 \leftarrow \mathcal{E}'_1 - e$
10:            $\mathcal{E}'_2 \leftarrow \mathcal{E}'_2 - e'$
11:         **end if**
12:     **end for**
13: **until** No more aligned entities are added to $\mathcal{S}$

Table 4: Statistics of MMKGs in the dataset.

| Statistics | FB15K | DB15K | YG15K |
|---|---|---|---|
| # Entities | 14,951 | 12,842 | 15,404 |
| # Relations | 1,345 | 279 | 32 |
| # Attributes | 116 | 225 | 7 |
| # Relational Triples | 592,213 | 89,197 | 122,886 |
| # Attribute Triples | 29,395 | 48,080 | 23,532 |
| # Entity-Image Pairs | 13,444 | 12,837 | 11,194 |
| # Aligned Entity Pairs | - | 12,846 | 11,199 |

appears within the top-N predicted entities and calculates the average across all source entities. MRR denotes the mean reciprocal rank of correctly aligned entities, and MR represents the mean rank of correctly aligned entities. A higher value for H@N and MRR indicates better performance, whereas a lower value for MR suggests the same. The formulas for calculating these metrics are as follows:

$$\text{H@N} = \frac{1}{|\mathcal{S}|} \sum_{i=1}^{|\mathcal{S}|} \mathbb{I}\left[\text{rank}_i \leq \text{N}\right], \tag{21}$$

$$\text{MRR} = \frac{1}{|\mathcal{S}|} \sum_{i=1}^{|\mathcal{S}|} \frac{1}{\text{rank}_i}, \text{MR} = \frac{1}{|\mathcal{S}|} \sum_{i=1}^{|\mathcal{S}|} \text{rank}_i, \tag{22}$$

where $\mathcal{S}$ refers to the set of aligned entity pairs, $\text{rank}_i$ denotes the rank of the correctly aligned entity for the $i$-th query entity, and $\mathbb{I}\left[\cdot\right]$ is the function of indicator.

### C.3 Model Configurations

We initialized all weight matrices with Xavier Normal initializer [15]. For efficiency, we froze the parameters of pre-trained ViT [14] and BERT [11] in the training stage. The dimensions of all entity features and $\mathbf{r}$ were 100. In VAEs, the encoder and decoder were both composed of two fully connected layers, and the dimension of the latent representations was set to 64. The number of heads $\eta$ is 2. In $\mathcal{L}_{TransE}$, we set $\gamma$ as 1. In $\mathcal{L}_{cl}$, $\gamma_{cl}$ was 2. In the overall objective, $\lambda_1$ and $\lambda_2$ were 1e-2. We used the mini-batch method with a batch size of 5000. The learning rate was 0.001 and Adam optimizer [17] was adopted. Our model was implemented using the framework of PyTorch [7]. The experiments were conducted on a server with two Intel Xeon Silver 4214R CPUs @ 2.40GHz, four NVIDIA GeForce RTX 3090 GPUs, and 256 GB RAM memory.

### C.4 Baseline Description

We selected both representative and state-of-the-art baselines for performance comparison. The descriptions of these baselines are introduced in detail as follows:

**For traditional KGs**:

- **IPTransE** [53] is the first to propose an iterative strategy for entity alignment to iteratively create newly aligned entity pairs for soft alignment.

- **GCN-Align** [36] leverages the graph convolutional networks to aggregate structural and attribute knowledge of entities.

- **BootEA** [31] adopts the iterative strategy for data augmentation and designs an alignment editing method to reduce error accumulation during the process of iterations.

- **SEA** [26] exploits labeled data and abundant unlabeled data together and learns the difference in degree of nodes with adversarial training.

- **RAC** [46] contrasts different views of entity representations by designing an unsupervised contrastive loss and exploits the unlabeled data to augment supervision signals.

- **PEEA** [32] increases the connections between far-away entities and labels ones by incorporating positional information into the representation learning with a position attention layer for fine-grained information.

- **MixTEA** [42] instructs the model learning by an end-to-end mixture teaching of manually labeled mappings and probabilistic pseudo mappings.

**For MMKGs**:

- **PoE** [23] employs a Product of Experts approach to calculate the probability of alignment following the generation of individual uni-modal representations, subsequently determining the "SameAs" links between aligned entities.

- **MMEA** [5] initially produces entity representations with relational, visual, and numerical features and transfers each uni-modal feature embedding into a unified space to facilitate integration.

- **EVA** [22] adopts ResNet, GCN, feedforward network to learn images, graph structure, and triples, and then incorporates multi-modal data into a combined embedding using an attention-driven modality weighting.

- **MSNEA** [6] implements inter-modal enhancement that leverages visual features to steer the learning of relational and attribute features, with multi-modal contrastive learning to mitigate the overpowering impact of weak modalities.

- **MCLEA** [21] takes into account task-specific modalities and models the inter-modal relationships for each entity representation. It then employs contrastive learning to jointly capture both intra-modal and inter-modal interactions.

- **ACK-MMEA** [19] integrates consistent alignment knowledge via merge and generate operators, and leverages a graph neural network called ConsistGNN to aggregate consistent information.

- **MoAlign** [20] hierarchically incorporates neighbor features, multi-modal attributes, and entity types by a hierarchical modifiable self-attention block in a transformer encoder to maintain the unique semantics of different information.

The results of IPTransE, GCN-Align, SEA, PoE, and MMEA are reported in [5]. The results of BootEA, RAC, EVA, and MSNEA are reported in [6]. The results of ACK-MMEA and MoAlign are reported in [19] and [20]. The results of PEEA [8], MixTEA [9], and MCLEA [10] are reproduced by running the codes in GitHub with default hyperparameter settings.

## C.5 Parameter Sensitivity

We have introduced coefficients $\lambda_1$ and $\lambda_2$ for the two constraint losses in the overall objective function to balance the impact of different losses. Here, we investigate the performance change of TMEA with varying coefficients. We conducted experiments by choosing values for $\lambda_1$ and $\lambda_2$ from the list [1e-5, 1e-4, 1e-3, 1e-2, 1e-1, 1], and the results on FB15K-DB15K are illustrated in Figure 5. The results indicate that the optimal values for $\lambda_1$ and $\lambda_2$ are 1e-2. When values are greater than 1e-2, the performance shows a declining trend. This is because excessive emphasis on constraint behavior can lead to a deviation in the optimization direction.

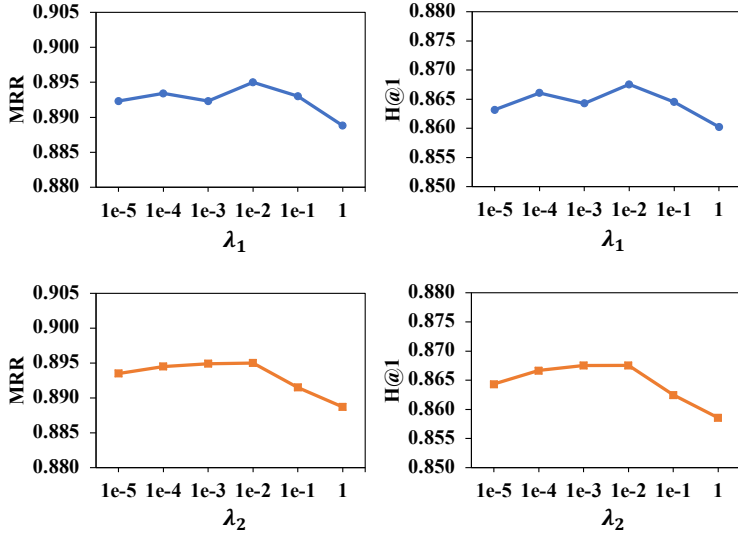

Figure 5: The parameter sensitivity on FB15K-DB15K.

# D  Broader Impact

The impact of TMEA extends beyond its immediate application in integrating knowledge graphs from diverse data sources. By addressing the challenges of tackling uncertain correspondences, TMEA offers a framework that can be generalized to various multimodal learning tasks like multimodal recommendation systems, multimodal document classification, and multimodal sentiment analysis.

For instance, in the domain of multi-modal recommendation systems, where diverse sources of information need to be fused to provide personalized recommendations, TMEA can be adapted to enhance the alignment between different modalities (e.g., user preferences, item descriptions, and user-item interactions). By leveraging its alignment-augmented abstract representation and inter-modal commonality enhancement mechanism, TMEA can effectively capture the subtle semantic associations between modalities, leading to more accurate and personalized recommendations.

Furthermore, TMEA's approach to handling diverse attribute knowledge descriptions and modality absence can also benefit other multi-modal learning tasks, such as image-text matching, cross-modal retrieval, and multimodal sentiment analysis. In these tasks, where aligning and integrating information from multiple modalities is crucial, TMEA's methodology of unifying modality features and enhancing commonalities can contribute to improved performance and robustness.

By fostering an environment where multi-modal data is more accurately aligned and integrated, the TMEA model paves the way for novel applications and research opportunities. For instance, in healthcare, it could support more robust diagnostic tools and patient care strategies by providing a comprehensive view of patient data across modalities. In financial markets, it might offer deeper insights through the integration of textual reports, market data, and socio-economic indicators, leading to more informed investment strategies. Beyond these, the model's application could extend to areas like intelligent urban planning, environmental monitoring, and personalized education, each benefiting from the nuanced understanding and integration of diverse data streams.

In summary, TMEA's broader impact lies in its potential to advance the field of multi-modal learning by providing a versatile framework that can be applied to a wide range of tasks or scenarios requiring multi-modal fusion and alignment.

## E Limitations

Although the TMEA method offers apparent advantages, its practical application is limited by several factors. Firstly, the attribute knowledge encoder relies on the large language model trained on general corpora, rather than KG-specific data. Consequently, the model's semantic understanding may lack the depth and specificity necessary for KG contexts. In the future, we will explore fine-tuning open-source large language models to enhance semantic understanding specifically tailored to KG attributes. Secondly, TMEA depends on aligned entity annotation data for feature extraction and model optimization. This dependency presents challenges, particularly in scenarios where annotations are scarce or costly to obtain. To address this issue, our future investigations will focus on fully leveraging multi-modal data in unsupervised settings, aiming to mitigate the reliance on annotated data while enhancing model performance.

